# Patient Risk Stratification for Hospital-Associated C. *diff* as a Time-Series Classification Task

**Jenna Wiens**
jwiens@mit.edu

**John V. Guttag**
guttag@mit.edu

**Eric Horvitz**
horvitz@microsoft.com

## Abstract

A patient's risk for adverse events is affected by temporal processes including the nature and timing of diagnostic and therapeutic activities, and the overall evolution of the patient's pathophysiology over time. Yet many investigators ignore this temporal aspect when modeling patient outcomes, considering only the patient's current or aggregate state. In this paper, we represent patient risk as a time series. In doing so, patient risk stratification becomes a time-series classification task. The task differs from most applications of time-series analysis, like speech processing, since the time series itself must first be extracted. Thus, we begin by defining and extracting approximate *risk processes*, the evolving approximate daily risk of a patient. Once obtained, we use these signals to explore different approaches to time-series classification with the goal of identifying high-risk patterns. We apply the classification to the specific task of identifying patients at risk of testing positive for hospital acquired Clostridium *difficile*. We achieve an area under the receiver operating characteristic curve of 0.79 on a held-out set of several hundred patients. Our two-stage approach to risk stratification outperforms classifiers that consider only a patient's current state ($p < 0.05$).

## 1  Introduction

Time-series data are available in many different fields, including medicine, finance, information retrieval and weather prediction. Much research has been devoted to the analysis and classification of such signals [1] [2]. In recent years, researchers have had great success with identifying temporal patterns in such time series and with methods that forecast the value of variables. In most applications there is an explicit time series, e.g., ECG signals, stock prices, audio recordings, or daily average temperatures.

We consider a novel application of time-series analysis, patient risk. Patient risk has an inherent temporal aspect; it evolves over time as it is influenced by intrinsic and extrinsic factors. However, it has no easily measurable time series. We hypothesize that, if one could measure risk over time, one could learn patterns of risk that are more likely to lead to adverse outcomes. In this work, we frame the problem of identifying hospitalized patients for high-risk outcomes as a time-series classification task. We propose and motivate the study of patient *risk processes* to model the evolution of risk over the course of a hospital admission.

Specifically, we consider the problem of using time-series data to estimate the risk of an inpatient becoming colonized with Clostridium *difficile* (C. *diff*) during a hospital stay. (C. *diff* is a bacterial infection most often acquired in hospitals or nursing homes. It causes severe diarrhea and can lead to colitis and other serious complications.) Despite the fact that many of the risk factors are well known (e.g., exposure, age, underlying disease, use of antimicrobial agents, *etc.*) [3], C. *diff* continues to be a significant problem in many US hospitals. From 1996 to 2009, C. *diff* rates for hospitalized patients aged $\geq 65$ years increased by 200% [4].

There are well-established clinical guidelines for predicting whether a test for C. *diff* is likely to be positive [5]. Such guidelines are based largely on the presence of symptoms associated with an existing C. *diff* infection, and thus are not useful for predicting whether a patient will *become* infected. In contrast, risk stratification models aim to identify patients at high risk of becoming infected. The use of these models could lead to a better understanding of the risk factors involved and ultimately provide information about how to reduce the incidence of C. *diff* in hospitals.

There are many different ways to define the problem of estimating risk. The precise definition has important ramifications for both the potential utility of the estimate and the difficulty of the problem.

Reported results in the medical literature for the problem of risk stratification for C. *diff* vary greatly, with areas under the receiver operating characteristic curve (AUC) of 0.628-0.896 [6] [7][8][9][10]. The variation in classification performance is based in part on differences in the task definition, in part on differences in the study populations, and in part on the evaluation method. The highest reported AUCs were from studies of small (e.g., 50 patients) populations, relatively easy tasks (e.g., inclusion of large number of patients with predictably short stays, e.g., patients in labor), or both. Additionally, some of the reported results were not obtained from testing on held-out sets.

We consider patients with at least a 7-day hospital admission who do not test positive for C. *diff* until day 7 or later. This group of patients is already at an elevated risk for acquiring C. *diff* because of the duration of the hospital stay. Focusing on this group makes the problem more relevant (and more difficult) than other related tasks.

To the best of our knowledge, representing and studying the risk of acquiring C. *diff* (or any other infection) as a time series has not previously been explored. We propose a risk stratification method that aims to identify patterns of risk that are more likely to lead to adverse outcomes. In [11] we proposed a method for extracting patient risk processes. Once patient risk processes are extracted, the problem of risk stratification becomes that of time-series classification. We explore a variety of different methods including classification using similarity metrics, feature extraction, and hidden Markov models. A direct comparison with the reported results in the literature for C. *diff* risk prediction is difficult because of the differences in the studies mentioned above. Thus, to measure the added value of considering the temporal dimension, we implemented the standard approach as represented in the related literature of classifying patients based on their current or average state and applied it to our data set. Our method leads to a significant improvement over this more traditional approach.

## 2  The Data

Our dataset comes from a large US hospital database. We extracted all stays $>= 7 days$, from all inpatient admissions that occurred over the course of a year.

To ensure that we are in fact predicting the acquisition of C. *diff* during the current admission, we remove patients who tested positive for C. *diff* in the 60 days preceding or, if negative, following the current admission [3]. In addition, we remove patients who tested positive for C. *diff* before day 7 of the admission. Positive cases are those patients who test positive on or after 7 days in the hospital. Negative patients are all remaining patients.

We define the start of the risk period of a patient as the time of admission and define the end of the risk period, according to the following rule: if the patient tests positive, the first positive test marks the end of the risk period, otherwise the patient is considered at risk until discharge. The final population consisted of 9,751 hospital admissions and 8,166 unique patients. Within this population, 177 admissions had a positive test result for C. *diff*.

## 3  Methods

Patient risk is not a directly measurable time series. Thus, we propose a two-stage approach to risk stratification. We first extract approximate risk processes and then apply time-series classification techniques to those processes. Both stages are described here; for more detail regarding the first stage we direct the reader to [11].

## 3.1 Extracting Patient Risk Processes

We extract approximate patient risk processes, i.e., a risk time series for each admission, by independently calculating the daily risk of a patient and then concatenating these predictions. We begin by extracting more than 10,000 variables for each day of each hospital admission. Almost all of the features pertain to categorical features that have been exploded into binary features; hence the high dimensionality. Approximately half of the features are based on data collected at the time of admission e.g., patient history, admission reason, and patient demographics. These features remain constant throughout the stay. The remaining features are collected over the course of the admission and may change on a daily basis e.g., lab results, room location, medications, and vital sign measurements.

We employ a support vector machine (SVM) to produce daily risk scores. Each day of an admission is associated with its own feature vector. We refer to this feature vector of observations as the patient's current state. However, we do not have ground-truth labels for each day of a patient's admission. We only know whether or not a patient eventually tests positive for C. *diff*. Thus we assign each day of an admission in which the patient eventually tests positive as positive, even though the patient may not have actually been at high risk on each of those days. In doing so, we hope to identify high-risk patients as early as possible. Since we do not expect a patient's risk to remain constant during an entire admission, there is noise in the training labels. For example, there may be some days that look almost identical in the feature space but have different labels. To handle this noise we use a soft-margin SVM, that allows for misclassifications. As long as our assumption does not lead to more incorrect labels than correct labels, it is possible to learn a meaningful classifier, despite the approximate labels. We do not use the SVM as a classifier but instead consider the continuous prediction made by the SVM, i.e., the distance to the decision boundary. We take the concatenated continuous outputs of the SVM for a hospital admission as a representation of the approximate risk process. We give some examples of these approximate risk processes for both case and non-case patients in Figure 1.

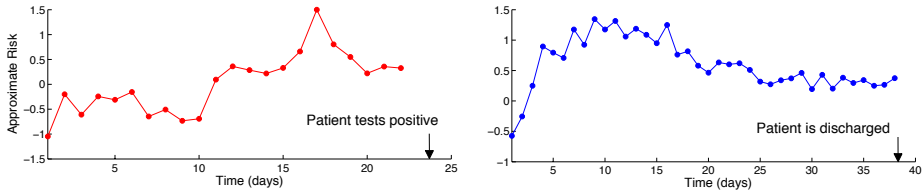

Figure 1: Approximate daily risk represented as a time series results in a *risk process* for each patient.

One could risk stratify patients based solely on their current state, i.e., use the daily risk value from the risk process to classify patients as either high risk or low risk on that day. This method, which ignores the temporal evolution of risk, achieves an AUC of 0.69 (95% CI 0.61-0.77). Intuitively, current risk should depend on previous risk. We tested this intuition by classifying patients based on the average of their risk process. This performed significantly better achieving an AUC of 0.75 (95% CI 0.69-0.81). Still, averaging in this way ignores the possibility of leveraging richer temporal patterns, as discussed in the next section.

## 3.2 Classifying Patient Risk Processes

Given the risk processes of each patient, the risk stratification task becomes a time-series classification task. Time-series classification is a well-investigated area of research, with many proposed methods. For an in-depth review of sequence classification we refer the reader to [2]. Here, we explore three different approaches to the problem: classification based on feature vectors, similarity measures, and finally HMMs. We first describe each method, and then present results about their performance in Section 4.

### 3.2.1 Classification using Feature Extraction

There are many different ways to extract features from time series. In the literature many have proposed time-frequency representations extracted using various Fourier or wavelet transforms [12]. Given the small number of samples composing our time-series data, we were wary of applying such techniques. Instead we chose an approach inspired by the combination of classifiers in the text domain using reliability indicators [13]. We define a feature vector based on different combinations of the predictions made in the first stage. We list the features in Table 1.

Table 1: Univariate summary statistics for observation vector $\mathbf{x} = [\mathbf{x_1}, \mathbf{x_2}, ..., \mathbf{x_n}]$

| Feature | Description | |
|---|---|---|
| 1 | length of time series | $n,$ |
| 2 | average daily risk | $\frac{1}{n}\sum_1^n x_i,$ |
| 3 | linear weighted average daily risk | $\frac{2}{n(n+1)}\sum_1^n i x_i,$ |
| 4 | quadratic weighted average daily risk | $\frac{6}{n(n+1)(2n+1)}\sum_1^n i^2 x_i,$ |
| 5 | risk on most recently observed day | $x_n,$ |
| 6 | standard deviation of daily risk | $\sigma,$ |
| 7 | average absolute change in daily risk | $\frac{1}{n}\sum_1^{n-1}|x_i - x_{i+1}|,$ |
| 8 | average absolute change in 1st difference | $\frac{1}{n}\sum_1^{n-2}|x_i' - x_{i+1}'|,$ |
| 9 | fraction of the visit with positive risk score | $\frac{1}{n}\sum_1^n 1_{x_i > 0},$ |
| 10 | fraction of the visit with negative risk score | $\frac{1}{n}\sum_1^n 1_{x_i < 0},$ |
| 11 | sum of the risk over the most recent 3 days | $\sum_{n-2}^n x_i,$ |
| 12 | longest positive run (normalized) | |
| 13 | longest negative run (normalized) | |
| 14 | maximum observation | $\max_i x_i,$ |
| 15 | location of maximum (normalized) | $\frac{1}{n}\operatorname*{argmax}_i x_i,$ |
| 16 | minimum observation | $\min_i x_i,$ |
| 17 | location of minimum (normalized) | $\frac{1}{n}\operatorname*{argmin}_i x_i,$ |

Features 2-4 are averages; Features 3 and 4 weight days closer to the time of classification more heavily. Features 6-10 are different measures for the amount of fluctuation in the time series. Features 5 and 11 capture information about the most recent states of the patient. Features 12 and 13 identify runs in the data, i.e., periods of time where the patient is consistently at high or low risk. Finally, Features 14-17 summarize information regarding global maxima and minima in the approximate risk process.

Given these feature definitions, we map each patient admission risk process to a fixed-length feature vector. These summarization variables allow one to compare time series of different lengths, while still capturing temporal information, e.g., when the maximum risk occurs relative to the time of prediction. Given this feature space, one can learn a classifier to identify high-risk patients. This approach is described in Figure 2.

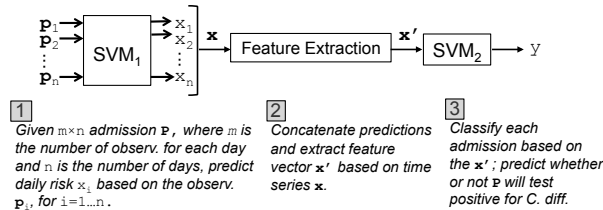

Figure 2: A two-step approach to risk stratification where predefined features are extracted from the time-series data.

### 3.2.2 Classification using Similarity Metrics

In the previous section, we learned a second classifier based on extracted features. In this section, we consider classifiers based on the raw data, i.e., the concatenated time series from Step 2 in Figure 2. SVMs classify examples based on a kernel or similarity measure. One of the most common non-linear kernels is the Gaussian radial basis function kernel: $k(\mathbf{x_i}, \mathbf{x_j}) = \exp(-\gamma \|\mathbf{x_i} - \mathbf{x_j}\|^2)$. Its output is dependent on the Euclidean distance between examples $\mathbf{x_i}$ and $\mathbf{x_j}$. This distance measure requires vectors of the same length. We consider two approaches to generating vectors of the same length: (1) linear interpolation and (2) truncation. In the first approach we linearly interpolate between points. In the second approach we consider only the most recent 5 days of data, $x_{n-4}, x_{n-3}, ..., x_n$.

Euclidean distance is a one-to-one comparison. In contrast, the dynamic time warping (DTW) distance is a one-to-many comparison [14]. DTW computes the distance between two time series by finding the minimal cost alignment. Here, the cost is the absolute distance between aligned points. We linearly interpolate all time series to have the same length, the length of the longest admission within the dataset (54). To ensure that the warping path does not contain lengthy vertical and horizontal segments, we constrain the warping window (how far the warping path can stray from the diagonal) using the Sakoe-Chiba band with a width of 10% of the length of the time series [15]. We learn an SVM classifier based on this distance metric, by replacing the Euclidean distance in the RBF kernel with the DTW distance, $k(\mathbf{x_i}, \mathbf{x_j}) = \exp(-\gamma DTW(\mathbf{x_i}, \mathbf{x_j}))$ as in [16].

### 3.2.3 Classification using Hidden Markov Models

We can make observations about a patient on a daily basis, but we cannot directly measure whether or not a patient is at high risk. Hence, we used the phrase *approximate* risk process. By applying HMMs we assume there is a sequence of hidden states, $x_1, x_2, ..., x_n$ that govern the observations $y_1, y_2, ..., y_n$. Here, the observations are the predictions made by the SVM. We consider a two-state HMM where each state, $s_1$ and $s_2$, is associated with a mixture of Gaussian distributions over possible observations. At an intuitive level, one can think of these states as representing low and high risk. Using the data, we learn and apply HMMs in two different ways.

**Classification via Likelihood**

We hypothesize that there may exist patterns of risk over time that are more likely to lead to a positive test result. To test this hypothesis, we first consider the classic approach to classification using HMMs described in Section VI-B [17]. We learn two separate HMMs: one using only observation sequences from positive patients and another using only observation sequences from negative patients. We initialize the emission probabilities differently for each model based on the data, but initialize the transition probabilities as uniform probabilities. Given a test observation sequence, we apply both models and calculate the log-likelihood of the data given each model using the forward-backward algorithm. We classify patients continuously, based on the ratio of the log-likelihoods.

**Classification via Posterior State Probabilities**

As we saw in Figure 1, the SVM output for a patient may fluctuate greatly from day to day. While large fluctuations in risk are not impossible, they are not common. Recall that in our initial calculation while the variables from time of admission are included in the prediction, the previous day's risk is not. The predictions produced by the SVM are independent. HMMs allow us to model the observations as a sequence and induce a temporal dependence in the model: the current state, $x_t$, depends on the previous state, $x_{t-1}$.

We learn an HMM on a training set. We consider a two state model in which we initialize the emission probabilities as $p(y_t|x_t = s_1) = N(\mu_{s1}, 1)$, $p(y_t|x_t = s_2) = N(\mu_{s2}, 1) \ \forall \ t$ where $\mu_{s1} = -1$ and $\mu_{s2} = 1$. Based on this initialization $s_1$ and $s_2$ correspond to "low-risk" and "high-risk" states, as mentioned above. A key decision was to use a left-to-right model where, once a patient reaches a "high-risk" state they remain there. All remaining transition probabilities were initialized uniformly. Applied to a test example we compute the posterior probabilities $p(x_t|y_1, ..., y_n)$ for $t = 1...n$ using the forward-backward algorithm. Because of the left-to-right assumption, if enough high-risk observations are made it will trigger a transition to the high-risk state. Figure 3 shows two examples of risk processes and their associated posterior state probabilities $p(x_t = s_2|y_1, ..., y_n)$ for

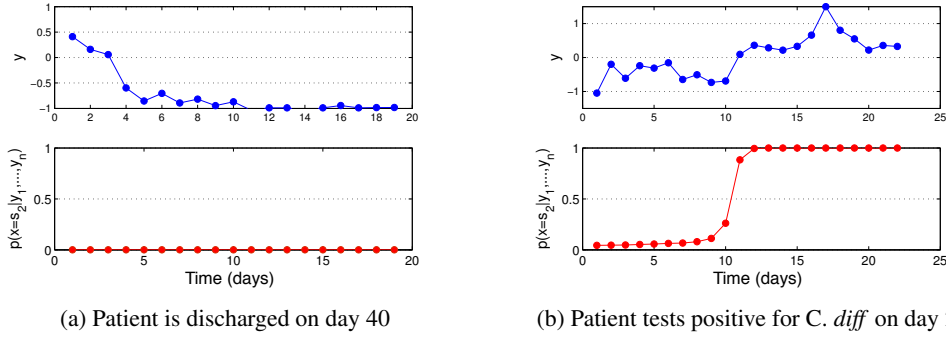

(a) Patient is discharged on day 40      (b) Patient tests positive for C. *diff* on day 24

Figure 3: Given all of the observations from $y_1, ..., y_n$ (in blue) we compute the posterior probability of being in a high-risk state for each day (in red).

$t = 1...n$. We classify each patient according to the probability of being in a high-risk state on the most recent day i.e., $p(x_n = s_2 | y_1, ...y_n)$.

## 4 Experiments & Results

This section describes a set of experiments used to compare several methods for predicting a patient's risk of acquiring C. *diff* during the current hospital admission. We start by describing the experimental setup, which is maintained across all experiments, and later present the results.

### 4.1 Experimental Setup

In order to reduce the possibility of confusing the risk of becoming colonized with C. *diff* with the existence of a current infection, for patients from the positive class we consider only data collected up to two days before a positive test result. This reduces the possibility of learning a classifier based on symptoms or treatment (a problem with some earlier studies).

For patients who never test positive, researchers typically use the discharge day as the index event [3]. However, this can lead to deceptively good results because patients nearing discharge are typically healthier than patients not nearing discharge. To avoid this problem, we define the index event for negative examples as either the halfway point of their admission, or 5 days into the admission, whichever is greater. We consider a minimum of 5 days for a negative patient since 5 days is the minimum amount of data we have for any positive patient (e.g., a patient who tests positive on day 7).

To handle class imbalance, we randomly subsample the negative class, selecting 10 negative examples for each positive example. When training the SVM we employ asymmetric cost parameters as in [18]. Additionally, we remove outliers, those patients with admissions longer than 60 days. Next, we randomly split the data into stratified training and test sets with a 70/30 split. The training set consisted of 1,251 admissions (127 positive), while the test set was composed of 532 admissions (50 positive). This split was maintained across all experiments. In all of the experiments, the training data was used for training purposes and validation of parameter selection, and the test set was used for evaluation purposes. For training and classification, we employed SVM$^{light}$ [19] and Kevin Murphy's HMM Toolbox [20].

### 4.2 Results

Table 2 compares the performance of eight different classifiers applied to the held-out test data. The first classifier is our baseline approach, described in Section 3.1, it classifies patients based solely on their current state. The second classifier *RP+Average* is an initial improvement on this approach, and classifies patients based on the average value of their risk process. The remaining classifiers are all based on time-series classification methods. *RP+Similarity$_{Euc.5days}$* classifies patients using a non-linear SVM based on the Euclidean distance between the most recent

Table 2: Predicting a positive test result two days in advance using different classifiers. *Current State* represents the traditional approach to risk stratification, and is the only classifier that is not based on patient Risk Processes (RP).

| Approach | AUC | 95% CI | F-Score | 95% CI |
|---|---|---|---|---|
| *Current State* | 0.69 | 0.61-0.77 | 0.28 | 0.19-0.38 |
| *RP+Average* | 0.75 | 0.69-0.81 | 0.32 | 0.21-0.41 |
| *RP+Similarity$_{Euc.5days}$* | 0.73 | 0.67-0.80 | 0.27 | 0.18-0.37 |
| *RP+HMM$_{likelihood}$* | 0.74 | 0.68-0.81 | 0.30 | 0.20-0.38 |
| *RP+Similarity$_{Euc.interp.}$* | 0.75 | 0.69-0.82 | 0.31 | 0.22-0.41 |
| *RP+Similarity$_{DTW}$* | 0.76 | 0.69-0.82 | 0.31 | 0.22-0.41 |
| *RP+HMM$_{posterior}$* | 0.76 | 0.70-0.82 | 0.30 | 0.21-0.41 |
| *RP+Features* | 0.79 | 0.73-0.85 | 0.37 | 0.24-0.49 |

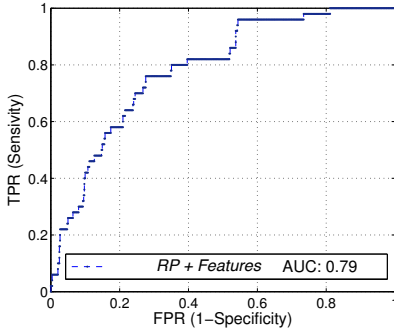

Figure 4: Results of predicting a patient's risk of testing positive for C. *diff* in the held-out test set using *RP+Features*.

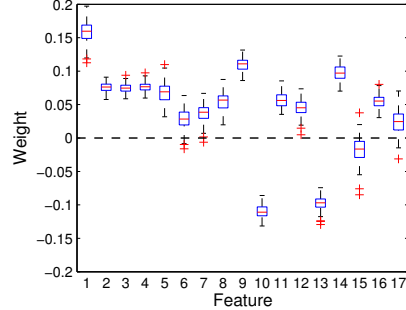

Figure 5: Feature weights from SVMs learned using different folds of the training set. The definition of features is given in Table 1

5 days. *RP+Similarity$_{Euc.interp.}$* uses the entire risk process by interpolating between points. These two methods in addition to *DTW* are described in Section 3.2.2. The difference between *RP+HMM$_{likelihood}$* and *RP+HMM$_{posterior}$* is described in Section 3.2.3. *RP+Features* classifies patients based on a linear combination of the average and other summary statistics (described in Section 3.2.1) of the risk process. For all of the performance measures we compute 95% point wise confidence intervals by bootstrapping (sampling with replacement) the held-out test set.

Figure 4 gives the ROC curve for the best method, the *RP+Features*. The AUC is calculated by sweeping the decision threshold. The *RP+Features* performed as well or better than the *Current State* and *RP+Average* approach at every point along the curve, thereby dominating both traditional approaches.

Compared to the other classifiers the classifier based on the *RP+Features* dominates on both AUC and F-Score. This classifier is based on a linear combination of statistics (listed in Table 1) computed from the patient risk processes. We learned the feature weights using the training data. To get a sense of the importance of each feature we used repeated sub-sampling validation on the training set. We randomly subsampled 70% of the training data 100 times and learned 100 different SVMs; this resulted in 100 different sets of feature weights. The results of this experiment are shown in Figure 5. The most important features are the length of the time series (Feature 1), the fraction of the time for which the patient is at positive risk (Feature 9), and the maximum risk attained (Feature 14). The only two features with significantly negative weights are Feature 10 and Feature 13, the overall fraction of time a patient has a negative risk, and the longest consecutive period of time that a patient has negative risk.

It is difficult to interpret the performance of a classifier based on these results alone, especially since the classes are imbalanced. Figure 6 gives the confusion matrix for mean performance of the best

classifier, *RP+Features*. To further convey the ability of the classifier to risk stratify patients, we split the test patients into quintiles (as is often done in clinical studies) based on the continuous output of the classifier. Each quintile contains approximately 106 patients. For each quintile we calculated the probability of a positive test result, based on those patients who eventually test positive for C. *diff*. Figure 7 shows that the probability increases with each quintile. The difference between the 1st and 5th quintiles is striking; relative to the 1st quintile, patients in the 5th quintile are at more than a 25-fold greater risk.

**Predicted Outcome**

|              |       | **p**  | **n**   |
|--------------|-------|--------|---------|
|              | **p′** | TP:26 | FN:24 |
| **Actual Outcome** |       |        |         |
|              | **n′** | FP:72 | TN:410 |

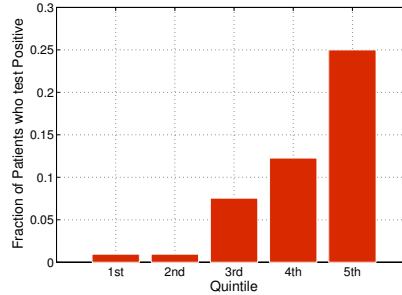

Figure 6: **Confusion Matrix** Using the best approach, the *RP+Features*, we achieve a Sensitivity of 50% and a Specificity of 85% on the held-out data.

Figure 7: Test patients with *RP+Features* predictions in the 5th quintile are more than 25 times more likely to test positive for C. *diff* than those in the 1st quintile.

## 5   Discussion & Conclusion

To the best of our knowledge, we are the first to consider risk of acquiring an infection as a time series. We use a two-stage process, first extracting approximate risk processes and then using the risk process as an input to a classifier. We explore three different approaches to classification: similarity metrics, feature vectors, and hidden Markov models. The majority of the methods based on time-series classification performed as well if not better than the previous approach of classifying patients simply based on the average of their risk process. The differences were not statistically significant, perhaps because of the small number of positive examples in the held-out set. Still, we are encouraged by these results, which suggest that posing the risk stratification problem as a time-series classification task can provide more accurate models.

There is large overlap in the confidence intervals for many of the results reported in Table 2, in part because of the paucity of positive examples. Still, based on the mean performance, all classifiers that incorporate patient risk processes outperform the *Current State* classifier, and the majority of those classifiers perform as well or better than the *RP+Average*. Only two classifiers did not perform better than the latter classifier: $RP+Similarity_{Euc.5days}$ and $RP+HMM_{likelihood}$. $RP+Similarity_{Euc.5days}$ classifies patients based on a similarity metric using only the most recent 5 days of the patient risk processes. Its relatively poor performance suggests that a patient's risk may depend on the entire risk process. The reasons for the relatively poor performance of the $RP+HMM_{likelihood}$ approach are less clear. Initially, we thought that perhaps two states was insufficient, but experiments with larger numbers of states led to overfitting on the training data. It may well be that the Markovian assumption is problematic in this context. We plan to investigate other graphical models, e.g., conditional random fields, going forward.

The F-Scores reported in Table 2 are lower than often seen in the machine-learning literature. However, when predicting outcomes in medicine, the problems are often so hard, the data so noisy, and the class imbalance so great that one cannot expect to achieve the kind of classification performance typically reported in the machine-learning literature. For this reason, the medical literature on risk stratification typically focuses on a combination of the AUC and the kind of odds ratios derivable from the data in Figure 7. As observed in the introduction, a direct comparison with the AUC achieved by others is not possible because of differences in the datasets, the inclusion criteria, and the details of the task. We have yet to thoroughly investigate the clinical ramifications of this work. However, for the daunting task of risk stratifying patients already at an elevated risk for C. *diff*, an AUC of 0.79 and an odds ratio of >25 are quite good.

# References

[1] M. M. Gaber, A. Zaslavsky, and S. Krishnaswamy. Mining data streams: A review. *SIGMOD*, 34(2), June 2005.

[2] Z. Xing, J. Pei, and E. Keogh. A brief survey on sequence classification. *ACM SIGKDD Explorations*, 12(1):40–48, June 2010.

[3] E. R. Dubberke, K. A. Reske, Y. Yan, M. A. Olsen, L. C. McDonald, and V. J. Fraser. Clostridium difficile - associated disease in a setting of endemicity: Identification of novel risk factors. *Clinical Infectious Diseases*, 45:1543–9, December 2007.

[4] CDC. Rates for clostridium difficile infection among hospitalized patients. *Centers for Disease Control and Prevention Morbidity and Mortality Weekly Report*, 60(34):1171, 2011.

[5] D. A. Katz, M.E. Lynch, and B. Littenber. Clinical prediction rules to optimize cytotoxin testing for clostridium difficile in hospitalized patients with diarrhea. *American Journal of Medicine*, 100(5):487–95, 1996.

[6] J. Tanner, D. Khan, D. Anthony, and J. Paton. Waterlow score to predict patietns at risk of developing clostridium difficile-associated disease. *Journal of Hospital Infection*, 71(3):239–244, 2009.

[7] E. R. Dubberke, Y. Yan, K. A. Reske, A.M. Butler, J. Doherty, V. Pham, and V.J. Fraser. Development and validation of a clostridium difficile infection risk prediction model. *Infect Control Hosp Epidemiol*, 32(4):360–366, 2011.

[8] K. W. Garey, T. K. Dao-Tran, Z. D. Jiang, M. P. Price, L. O. Gentry, and DuPont H. L. A clinical risk index for clostridium difficile infection in hospitalized patients receiving broad-spectrum antibiotics. *Journal of Hospital Infections*, 70(2):142–147, 2008.

[9] G. Krapohl. Preventing health care-associated infection: Development of a clinical prediction rule for clostridium difficile infection. PhD Thesis, 2011.

[10] N. Peled, S. Pitlik, Z. Samra, A. Kazakov, Y. Bloch, and J. Bishara. Predicting clostridium difficile toxin in hospitalized patients with antibiotic-associated diarrhea. *Infect Control Hosp Epidemiol*, 28(4):377–81, 2007.

[11] J. Wiens, E. Horvitz, and J. Guttag. Learning evolving patient risk processes for c. diff colonization. In *ICML Workshop on Machine Learning from Clinical Data*, 2012.

[12] T. W. Liao. Clustering of time series data - a survey. *The Journal of the Pattern Recognition Society*, January 2005.

[13] P. Bennett, S. Dumais, and E. Horvitz. The combination of test classifiers using reliability indicators. *Information Retrieval*, 8(1):67–100, 2005.

[14] H. Sakoe and S. Chiba. Dynamic programming algorithm optimization for spoken word recognition. *IEEE Transactions on Acoustics, Speech, and Signal Processing*, 26(1):43–49, 1978.

[15] C. Ratanamahatana and E. Keogh. Three myths about dynamic time warping data mining. In *Proceedings of the Fifth SIAM International Conference on Data Mining*, 2005.

[16] C. Bahlmann, B. Haasdonk, and Burkhardt H. On-line handwriting recognition with support vector machines - a kernel approach. *Proceedings of the 8th International Workshop on Frontiers in Handwriting Recognition*, 2002.

[17] L.R. Rabiner. A tutorial on hidden markov models and selected applications in speech recognition. *Proceedings of the IEEE*, 77(2), February 1989.

[18] K. Morik, P. Brockhausen, and T. Joachims. Combining statistical learning with a knowledge-based approach - a case study in intensive care monitoring. *Proc. 16th International Conference on Machine Learning*, 1999.

[19] T. Joachims. Making large-scale svm learning practical. advances in kernel methods - support vector learning, 1999.

[20] K. Murphy. Hidden Markov Model (HMM) Toolbox for Matlab. www.cs.ubc.ca/˜murphyk/Software/HMM/hmm.html.

